# Hierarchical Eigensolver for Transition Matrices in Spectral Methods

**Chakra Chennubhotla**[*] **and Allan D. Jepson**[†]
[*]Department of Computational Biology, University of Pittsburgh
[†]Department of Computer Science, University of Toronto

## Abstract

We show how to build hierarchical, reduced-rank representation for large stochastic matrices and use this representation to design an efficient algorithm for computing the largest eigenvalues, and the corresponding eigenvectors. In particular, the eigen problem is first solved at the coarsest level of the representation. The approximate eigen solution is then interpolated over successive levels of the hierarchy. A small number of power iterations are employed at each stage to correct the eigen solution. The typical speedups obtained by a Matlab implementation of our fast eigensolver over a standard sparse matrix eigensolver [13] are at least a factor of ten for large image sizes. The hierarchical representation has proven to be effective in a min-cut based segmentation algorithm that we proposed recently [8].

## 1 Spectral Methods

Graph-theoretic spectral methods have gained popularity in a variety of application domains: segmenting images [22]; embedding in low-dimensional spaces [4, 5, 8]; and clustering parallel scientific computation tasks [19]. Spectral methods enable the study of properties global to a dataset, using only local (pairwise) similarity or affinity measurements between the data points. The global properties that emerge are best understood in terms of a random walk formulation on the graph. For example, the graph can be partitioned into clusters by analyzing the perturbations to the stationary distribution of a Markovian relaxation process defined in terms of the affinity weights [17, 18, 24, 7]. The Markovian relaxation process need never be explicitly carried out; instead, it can be analytically expressed using the leading order eigenvectors, and eigenvalues, of the Markov transition matrix.

In this paper we consider the practical application of spectral methods to large datasets. In particular, the eigen decomposition can be very expensive, on the order of $O(n^3)$, where $n$ is the number of nodes in the graph. While it is possible to compute analytically the first eigenvector (see §3 below), the remaining subspace of vectors (necessary for say clustering) has to be explicitly computed. A typical approach to dealing with this difficulty is to first sparsify the links in the graph [22] and then apply an efficient eigensolver [13, 23, 3].

In comparison, we propose in this paper a specialized eigensolver suitable for large stochastic matrices with known stationary distributions. In particular, we exploit the spectral properties of the Markov transition matrix to generate hierarchical, successively lower-ranked approximations to the full transition matrix. The eigen problem is solved directly at the coarsest level of representation. The approximate eigen solution is then interpolated over successive levels of the hierarchy, using a small number of power iterations to correct the solution at each stage.

## 2 Previous Work

One approach to speeding up the eigen decomposition is to use the fact that the columns of the affinity matrix are typically correlated. The idea then is to pick a small number of representative columns to perform eigen decomposition via SVD. For example, in the Nystrom approximation procedure, originally proposed for integral eigenvalue problems, the idea is to randomly pick a small set of $m$ columns; generate the corresponding affinity matrix; solve the eigenproblem and finally extend the solution to the complete graph [9, 10]. The Nystrom method has also been recently applied in the kernel learning methods for fast Gaussian process classification and regression [25]. Other sampling-based approaches include the work reported in [1, 2, 11].

Our starting point is the transition matrix generated from affinity weights and we show how building a representational hierarchy follows naturally from considering the stochastic matrix. A closely related work is the paper by Lin on reduced rank approximations of transition matrices [14]. We differ in how we approximate the transition matrices, in particular our objective function is computationally less expensive to solve. In particular, one of our goals in reducing transition matrices is to develop a fast, specialized eigen solver for spectral clustering. Fast eigensolving is also the goal in ACE [12], where successive levels in the hierarchy can potentially have negative affinities. A graph coarsening process for clustering was also pursued in [21, 3].

## 3 Markov Chain Terminology

We first provide a brief overview of the Markov chain terminology here (for more details see [17, 15, 6]). We consider an undirected graph $G = (V, E)$ with vertices $v_i$, for $i = \{1, \ldots, n\}$, and edges $e_{i,j}$ with non-negative weights $a_{i,j}$. Here the weight $a_{i,j}$ represents the affinity between vertices $v_i$ and $v_j$. The affinities are represented by a non-negative, symmetric $n \times n$ matrix $A$ having weights $a_{i,j}$ as elements. The degree of a node $j$ is defined to be: $d_j = \sum_{i=1}^{n} a_{i,j} = \sum_{j=1}^{n} a_{j,i}$, where we define $D = \text{diag}(d_1, \ldots, d_n)$. A Markov chain is defined using these affinities by setting a transition probability matrix $M = AD^{-1}$, where the columns of $M$ each sum to 1. The transition probability matrix defines the random walk of a particle on the graph $G$.

The random walk need never be explicitly carried out; instead, it can be analytically expressed using the leading order eigenvectors, and eigenvalues, of the Markov transition matrix. Because the stochastic matrices need not be symmetric in general, a direct eigen decomposition step is not preferred for reasons of instability. This problem is easily circumvented by considering a normalized affinity matrix: $L = D^{-1/2}AD^{-1/2}$, which is related to the stochastic matrix by a similarity transformation: $L = D^{-1/2}MD^{1/2}$. Because $L$ is symmetric, it can be diagonalized: $L = U\Lambda U^T$, where $U = [\vec{u}_1, \vec{u}_2, \cdots, \vec{u}_n]$ is an orthogonal set of eigenvectors and $\Lambda$ is a diagonal matrix of eigenvalues $[\lambda_1, \lambda_2, \cdots, \lambda_n]$ sorted in decreasing order. The eigenvectors have unit length $\|\vec{u}_k\| = 1$ and from the form of $A$ and $D$ it can be shown that the eigenvalues $\lambda_i \in (-1, 1]$, with at least one eigenvalue equal to one. Without loss of generality, we take $\lambda_1 = 1$. Because $L$ and $M$ are similar we can perform an eigen decomposition of the Markov transition matrix as: $M = D^{1/2}LD^{-1/2} = D^{1/2}U \Lambda U^T D^{-1/2}$. Thus an eigenvector $\vec{u}$ of $L$ corresponds to an eigenvector $D^{1/2}\vec{u}$ of $M$ with the same eigenvalue $\lambda$.

The Markovian relaxation process after $\beta$ iterations, namely $M^\beta$, can be represented as: $M^\beta = D^{1/2}U\Lambda^\beta U^T D^{-1/2}$. Therefore, a particle undertaking a random walk with an initial distribution $\vec{p}^0$ acquires after $\beta$ steps a distribution $\vec{p}^\beta$ given by: $\vec{p}^\beta = M^\beta \vec{p}^0$. Assuming the graph is connected, as $\beta \to \infty$, the Markov chain approaches a unique stationary distribution given by $\vec{\pi} = \text{diag}(D)/\sum_{i=1}^{n} d_i$, and thus, $M^\infty = \vec{\pi}\mathbf{1}^T$, where $\mathbf{1}$ is a $n$-dim column vector of all ones. Observe that $\vec{\pi}$ is an eigenvector of $M$ as it is easy to show that $M\vec{\pi} = \vec{\pi}$ and the corresponding eigenvalue is 1. Next, we show how to generate hierarchical, successively low-ranked approximations for the transition matrix $M$.

# 4 Building a Hierarchy of Transition Matrices

The goal is to generate a very fast approximation, while simultaneously achieving sufficient accuracy. For notational ease, we think of $M$ as a fine-scale representation and $\widetilde{M}$ as some coarse-scale approximation to be derived here. By coarsening $\widetilde{M}$ further, we can generate successive levels of the representation hierarchy. We use the stationary distribution $\vec{\pi}$ to construct a corresponding coarse-scale stationary distribution $\vec{\delta}$. As we just discussed a critical property of the fine scale Markov matrix $M$ is that it is similar to the symmetric matrix $L$ and we wish to preserve this property at every level of the representation hierarchy.

## 4.1 Deriving Coarse-Scale Stationary Distribution

We begin by expressing the stationary distribution $\vec{\pi}$ as a probabilistic mixture of latent distributions. In matrix notation, we have

$$\vec{\pi} = K\vec{\delta}, \tag{1}$$

where $\vec{\delta}$ is an unknown mixture coefficient vector of length $m$, $K$ is an $n \times m$ non-negative kernel matrix whose columns are latent distributions that each sum to 1: $\sum_{i=1}^{n} K_{i,j} = 1$ and $m \ll n$. It is easy to derive a maximum likelihood approximation of $\vec{\delta}$ using an EM type algorithm [16]. The main step is to find a stationary point $\left(\vec{\delta}, \lambda\right)$ for the Lagrangian:

$$E \equiv -\sum_{i=1}^{n} \pi_i \ln \sum_{j=1}^{m} K_{i,j}\delta_j + \lambda\left(\sum_{j=1}^{m} \delta_j - 1\right). \tag{2}$$

An implicit step in this EM procedure is to compute the the ownership probability $r_{i,j}$ of the $j^{\text{th}}$ kernel (or node) at the coarse scale for the $i^{\text{th}}$ node on the fine scale and is given by

$$r_{i,j} = \frac{\delta_j K_{i,j}}{\sum_{k=1}^{m} \delta_k K_{i,k}}. \tag{3}$$

The EM procedure allows for an update of both $\vec{\delta}$ and the latent distributions in the kernel matrix $K$ (see §8.3.1 in [6]).

For initialization, $\vec{\delta}$ is taken to be uniform over the coarse-scale states. But in choosing kernels $K$, we provide a good initialization for the EM procedure. Specifically, the Markov matrix $M$ is diffused using a small number of iterations to get $M^\beta$. The diffusion causes random walks from neighboring nodes to be less distinguishable. This in turn helps us select a small number of columns of $M^\beta$ in a fast and greedy way to be the kernel matrix $K$. We defer the exact details on kernel selection to a later section (§4.3).

## 4.2 Deriving the Coarse-Scale Transition Matrix

In order to define $\widetilde{M}$, the coarse-scale transition matrix, we break it down into three steps. First, the Markov chain propagation at the coarse scale can be defined as:

$$\vec{q}^{\,k+1} = \widetilde{M}\vec{q}^{\,k}, \tag{4}$$

where $\vec{q}^{\,k}$ is the coarse scale probability distribution after $k$ steps of the random walk. Second, we *expand* $\vec{q}^{\,k}$ into the fine scale using the kernels $K$ resulting in a fine scale probability distribution $\vec{p}^{\,k}$:

$$\vec{p}^{\,k} = K\vec{q}^{\,k}. \tag{5}$$

Finally, we *lift* $\vec{p}^{\,k}$ back into the coarse scale by using the ownership probability of the $j^{\text{th}}$ kernel for the $i^{\text{th}}$ node on the fine grid:

$$q_j^{k+1} = \sum_{i=1}^{n} r_{i,j} p_i^{\,k} \tag{6}$$

Substituting for Eqs.(3) and (5) in Eq. 6 gives

$$q_j^{k+1} = \sum_{i=1}^n r_{i,j} \sum_{t=1}^m K_{i,t} q_t^k = \sum_{i=1}^n \left( \frac{\delta_j K_{i,j}}{\sum_{k=1}^m \delta_k K_{i,k}} \right) \sum_{t=1}^m K_{i,t} q_t^k. \tag{7}$$

We can write the preceding equation in a matrix form:

$$\vec{q}^{k+1} = \mathrm{diag}(\vec{\delta})\, K^T \,\mathrm{diag}\left( K\vec{\delta} \right)^{-1} K\vec{q}^k. \tag{8}$$

Comparing this with Eq. 4, we can derive the transition matrix $\widetilde{M}$ as:

$$\widetilde{M} = \mathrm{diag}(\vec{\delta})\, K^T \,\mathrm{diag}\left( K\vec{\delta} \right)^{-1} K. \tag{9}$$

It is easy to see that $\vec{\delta} = \widetilde{M}\vec{\delta}$, so $\vec{\delta}$ is the stationary distribution for $\widetilde{M}$. Following the definition of $\widetilde{M}$, and its stationary distribution $\vec{\delta}$, we can generate a *symmetric* coarse scale affinity matrix $\widetilde{A}$ given by

$$\widetilde{A} = \widetilde{M}\mathrm{diag}(\vec{\delta}) = \left( \mathrm{diag}(\vec{\delta})\, K^T \right) \left( \mathrm{diag}\left( K\vec{\delta} \right)^{-1} \right) \left( K\mathrm{diag}(\vec{\delta}) \right), \tag{10}$$

where we substitute for the expression $\widetilde{M}$ from Eq. 9. The coarse-scale affinity matrix $\widetilde{A}$ is then normalized to get:

$$\widetilde{L} = \widetilde{D}^{-1/2}\widetilde{A}\widetilde{D}^{-1/2}; \quad \widetilde{D} = \mathrm{diag}(\widetilde{d}_1, \widetilde{d}_2, \cdots, \widetilde{d}_m), \tag{11}$$

where $\widetilde{d}_j$ is the degree of node $j$ in the coarse-scale graph represented by the matrix $\widetilde{A}$ (see §3 for degree definition). Thus, the coarse scale Markov matrix $\widetilde{M}$ is precisely similar to a symmetric matrix $\widetilde{L}$.

### 4.3 Selecting Kernels

For demonstration purpose, we present the kernel selection details on the image of an eye shown below. To begin with, a random walk is defined where each pixel in the test image is associated with a vertex of the graph $G$. The edges in $G$ are defined by the standard 8-neighbourhood of each pixel. For the demonstrations in this paper, the edge weight $a_{i,j}$ between neighbouring pixels $x_i$ and $x_j$ is given by a function of the difference in the corresponding intensities $I(x_i)$ and $I(x_j)$: $a_{i,j} = \exp(-(I(x_i) - I(x_j))^2/2\sigma_a^2)$, where $\sigma_a$ is set according to the median absolute difference $|I(x_i) - I(x_j)|$ between neighbours measured over the entire image. The affinity matrix $A$ with the edge weights is then used to generate a Markov transition matrix $M$.

The kernel selection process we use is fast and greedy. First, the fine scale Markov matrix $M$ is diffused to $M^\beta$ using $\beta = 4$. The Markov matrix $M$ is sparse as we make the affinity matrix $A$ sparse. Every column in the diffused matrix $M^\beta$ is a potential kernel. To facilitate the selection process, the second step is to rank order the columns of $M^\beta$ based on a probability value in the stationary distribution $\vec{\pi}$. Third, the kernels (i.e. columns of $M^\beta$) are picked in such a way that for a kernel $K_i$ all of the neighbours of pixel $i$ which are within the *half-height* of the the maximum value in the kernel $K_i$ are suppressed from the selection process. Finally, the kernel selection is continued until every pixel in the image is within a half-height of the peak value of at least one kernel. If $M$ is a full matrix, to avoid the expense of computing $M^\beta$ explicitly, random kernel centers can be selected, and only the corresponding columns of $M^\beta$ need be computed.

We show results from a three-scale hierarchy on the eye image (below). The image has $25 \times 20$ pixels but is shown here enlarged for clarity. At the first coarse scale 83 kernels are picked. The kernels each correspond to a different column in the fine scale transition matrix and the pixels giving rise to these kernels are shown numbered on the image.

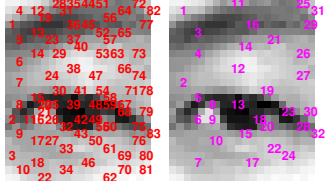

Coarse Scale 1    Coarse Scale 2

Using these kernels as an initialization, the EM procedure derives a coarse-scale stationary distribution $\vec{\delta}$ (Eq. 2), while simultaneously updating the kernel matrix. Using the newly updated kernel matrix $K$ and the derived stationary distribution $\vec{\delta}$ a transition matrix $\widetilde{M}$ is generated (Eq. 9). The coarse scale Markov matrix is then diffused to $\widetilde{M}^{\beta}$, again using $\beta = 4$. The kernel selection algorithm is reapplied, this time picking 32 kernels for the second coarse scale. Larger values of $\beta$ cause the coarser level to have fewer elements. But the exact number of elements depends on the form of the kernels themselves. For the random experiments that we describe later in §6 we found $\beta = 2$ in the first iteration and 4 thereafter causes the number of kernels to be reduced by a factor of roughly $1/3$ to $1/4$ at each level.

At coarser levels of the hierarchy, we expect the kernels to get less sparse and so will the affinity and the transition matrices. In order to promote sparsity at successive levels of the hierarchy we sparsify $\widetilde{A}$ by zeroing out elements associated with "small" transition probabilities in $\widetilde{M}$. However, in the experiments described later in §6, we observe this sparsification step to be not critical. To summarize, we use the stationary distribution $\vec{\pi}$ at the fine-scale to derive a transition matrix $\widetilde{M}$, and its stationary distribution $\vec{\delta}$, at the coarse-scale. The coarse scale transition in turn helps to derive an affinity matrix $\widetilde{A}$ and its normalized version $\widetilde{L}$. It is obvious that this procedure can be repeated recursively. We describe next how to use this representation hierarchy for building a fast eigensolver.

## 5   Fast EigenSolver

Our goal in generating a hierarchical representation of a transition matrix is to develop a fast, specialized eigen solver for spectral clustering. To this end, we perform a full eigen decomposition of the normalized affinity matrix only at the coarsest level. As discussed in the previous section, the affinity matrix at the coarsest level is not likely to be sparse, hence it will need a full (as opposed to a sparse) version of an eigen solver. However it is typically the case that $e \leq m \ll n$ (even in the case of the three-scale hierarchy that we just considered) and hence we expect this step to be the least expensive computationally. The resulting eigenvectors are interpolated to the next lower level of the hierarchy by a process which will be described next. Because the eigen interpolation process between every adjacent pair of scales in the hierarchy is similar, we will assume we have access to the leading eigenvectors $\widetilde{U}$ (size: $m \times e$) for the normalized affinity matrix $\widetilde{L}$ (size: $m \times m$) and describe how to generate the leading eigenvectors $U$ (size: $n \times e$), and the leading eigenvalues $S$ (size: $e \times 1$), for the fine-scale normalized affinity matrix $L$ (size: $n \times n$). There are several steps to the eigen interpolation process and in the discussion that follows we refer to the lines in the pseudo-code presented below.

First, the coarse-scale eigenvectors $\widetilde{U}$ can be interpolated using the kernel matrix $K$ to generate $U = K\widetilde{U}$, an approximation for the fine-scale eigenvectors (line 9). Second, interpolation alone is unlikely to set the directions of $U$ exactly aligned with $U_L$, the vectors one would obtain by a direct eigen decomposition of the fine-scale normalized affinity matrix $L$. We therefore update the directions in $U$ by applying a small number of power iterations with $L$, as given in lines 13-15.

**function** $(U, S) = \text{CoarseToFine}(L, K, \widetilde{U}, \widetilde{S})$
1:  INPUT
2:     $L, K \Leftarrow \{L$ is $n \times n$ and $K$ is $n \times m$ where $m \ll n\}$
3:     $\widetilde{U}/\widetilde{S} \Leftarrow \{$leading coarse-scale eigenvectors/eigenvalues of $\widetilde{L}$. $\widetilde{U}$ is of size $m \times e, e \leq m\}$
4:  OUTPUT
5:     $U, S \Leftarrow \{$leading fine-scale eigenvectors/eigenvalues of $L$. $U$ is $n \times e$ and $S$ is $e \times 1.\}$

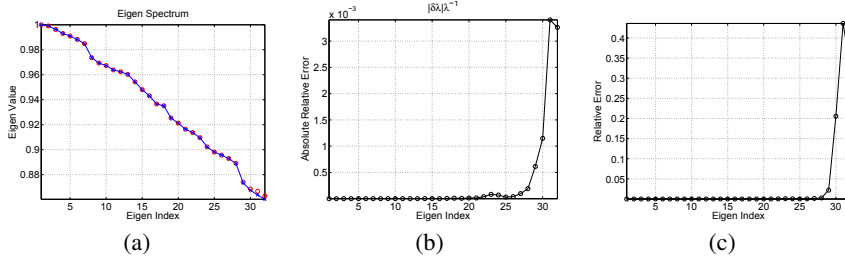

<center>(a)               (b)               (c)</center>

Figure 1: Hierarchical eigensolver results. (a) comparing ground truth eigenvalues $S_L$ (red circles) with multi-scale eigensolver spectrum $S$ (blue line) (b) Relative absolute error between eigenvalues: $\frac{|S - S_L|}{S_L}$ (c) Eigenvector mismatch: $\mathbf{1} - \mathrm{diag}\left(|U^T U_L|\right)$, between eigenvectors $U$ derived by the multi-scale eigensolver and the ground truth $U_L$. Observe the slight mismatch in the last few eigenvectors, but excellent agreement in the leading eigenvectors (see text).

```
 6: CONSTANTS:    TOL = 1e-4;    POWER_ITERS = 50
 7:
 8: TPI = min(POWER_ITERS, log(e × eps/TOL)/log(min(S̃)))  {eps: machine accuracy}
 9: U = KŨ {interpolation from coarse to fine}
10: while not converged do
11:     U_old = U {n × e matrix, e ≪ n}
12:     for i = 1 to TPI do
13:         U ⇐ LU
14:     end for
15:     U ⇐ Gram-Schmidt(U) {orthogonalize U}
16:     L_e = U^T LU {L may be sparse, but L_e need not be.}
17:     U_e S_e U_e^T = svd(L_e) {eigenanalysis of L_e, which is of size e × e.}
18:     U ⇐ UU_e {update the leading eigenvectors of L}
19:     S = diag(S_e) {grab the leading eigenvalues of L}
20:     innerProd = 1 − diag( U_old^T U ) {1 is a e × 1 vector of all ones}
21:     converged = max[abs(innerProd)] < TOL
22: end while
```

The number of power iterations TPI can be bounded as discussed next. Suppose $\vec{v} = Uc$ where $U$ is a matrix of true eigenvectors and $c$ is a coefficient vector for an arbitrary vector $\vec{v}$. After TPI power iterations $\vec{v}$ becomes $\vec{v} = U\mathrm{diag}(S^{\mathrm{TPI}})c$, where $S$ has the exact eigenvalues. In order for the component of a vector $\vec{v}$ in the direction $U_e$ (the $e^{\mathrm{th}}$ column of $U$) not to be swamped by other components, we can limit it's decay after TPI iterations as follows: $(S(e)/S(1))^{\mathrm{TPI}} >= e \times \mathrm{eps}/\mathrm{TOL}$, where $S(e)$ is the exact $e^{\mathrm{th}}$ eigenvalue, $S(1) = 1$, eps is the machine precision, TOL is requested accuracy. Because we do not have access to the exact value $S(e)$ at the beginning of the interpolation procedure, we estimate it from the coarse eigenvalues $\tilde{S}$. This leads to a bound on the power iterations TPI, as derived on the line 9 above. Third, the interpolation process and the power iterations need not preserve orthogonality in the eigenvectors in $U$. We fix this by Gram-Schmidt orthogonalization procedure (line 16). Finally, there is a still a problem with power iterations that needs to be resolved, in that it is very hard to separate nearby eigenvalues. In particular, for the convergence of the power iterations the ratio that matters is between the $(e + 1)^{\mathrm{st}}$ and $e^{\mathrm{th}}$ eigenvalues. So the idea we pursue is to use the power iterations only to separate the reduced space of eigenvectors (of dimension $e$) from the orthogonal subspace (of dimension $n - e$). We then use a full SVD on the reduced space to update the leading eigenvectors $U$, and eigenvalues $S$, for the fine-scale (lines 17-20). This idea is similar to computing the Ritz values and Ritz vectors in a Rayleigh-Ritz method.

## 6 Interpolation Results

Our multi-scale decomposition code is in Matlab. For the direct eigen decomposition, we have used the Matlab program `svds.m` which invokes the compiled ARPACKC routine [13], with a default convergence tolerance of `1e-10`.

In Fig. 1a we compare the spectrum $S$ obtained from a three-scale decomposition on the eye image (blue line) with the ground truth, which is the spectrum $S_L$ resulting from direct eigen decomposition of the fine-scale normalized affinity matrices $L$ (red circles). There is an excellent agreement in the leading eigenvalues. To illustrate this, we show absolute relative error between the spectra: $\frac{|S-S_L|}{S_L}$ in Fig. 1b. The spectra agree mostly, except for the last few eigenvalues. For a quantitative comparison between the eigenvectors, we plot in Fig. 1c the following measure: $\mathbf{1} - \mathrm{diag}(|U^T U_L|)$, where $U$ is the matrix of eigenvectors obtained by the multi-scale approximation, $U_L$ is the ground-truth resulting from a direct eigen decomposition of the fine-scale affinity matrix $L$ and $\mathbf{1}$ is a vector of all ones. The relative error plot demonstrates a close match, within the tolerance threshold of `1e-4` that we chose for the multi-scale method, in the leading eigenvector directions between the two methods. The relative error is high with the last few eigen vectors, which suggests that the power iterations have not clearly separated them from other directions. So, the strategy we suggest is to pad the required number of leading eigen basis by about 20% before invoking the multi-scale procedure. Obviously, the number of hierarchical stages for the multi-scale procedure must be chosen such that the transition matrix at the coarsest scale can accommodate the slight increase in the subspace dimensions. For lack of space we are omitting extra results (see Ch.8 in [6]).

Next we measure the time the hierarchical eigensolver takes to compute the leading eigen-basis for various input sizes, in comparison with the `svds.m` procedure [13]. We form images of different input sizes by Gaussian smoothing of i.i.d noise. The Gaussian function has a standard deviation of 3 pixels. The edges in graph $G$ are defined by the standard 8-neighbourhood of each pixel. The edge weights between neighbouring pixels are simply given by a function of the difference in the corresponding intensities (see §4.3). The affinity matrix $A$ with the edge weights is then used to generate a Markov transition matrix $M$. The fast eigensolver is run on ten different instances of the input image of a given size and the average of these times is reported here. For a fair comparison between the two procedures, we set the convergence tolerance value for the `svds.m` procedure to be `1e-4`, the same as the one used for the fast eigensolver. We found the hierarchical representation derived from this tolerance threshold to be sufficiently accurate for a novel min-cut based segmentation results that we reported in [8]. Also, the subspace dimensionality is fixed to be 51 where we expect (and indeed observe) the leading 40 eigenpairs derived from the multi-scale procedure to be accurate. Hence, while invoking `svds.m` we compute only the leading 41 eigenpairs.

In the table shown below, the first column corresponds to the number of nodes in the graph, while the second and third columns report the time taken in seconds by the `svds.m` procedure and the Matlab implementation of the multi-scale eigensolver respectively. The fourth column reports the speedups of the multi-scale eigensolver over `svds.m` procedure on a standard desktop (Intel P4, 2.5GHz, 1GB RAM). Lowering the tolerance threshold for `svds.m` made it faster by about $20 - 30\%$. Despite this, the multi-scale algorithm clearly outperforms the `svds.m` procedure. The most expensive step in the multi-scale algorithm is the power iteration required in the last stage, that is interpolating eigenvectors from the first coarse scale to the required fine scale. The complexity is of the order of $n \times e$ where $e$ is the subspace dimensionality and $n$ is the size of the graph. Indeed, from the table we can see that the multi-scale procedure is taking time roughly proportional to $n$. Deviations from the linear trend are observed at specific values of $n$, which we believe are due to the

| $n$ | `svds.m` | Multi-Scale | Speedup |
| --- | --- | --- | --- |
| $32^2$ | 1.6 | 1.5 | 1.1 |
| $63^2$ | 10.8 | 4.9 | 2.2 |
| $64^2$ | 20.5 | 5.5 | 3.7 |
| $65^2$ | 12.6 | 5.1 | 2.5 |
| $100^2$ | 44.2 | 13.1 | 3.4 |
| $127^2$ | 91.1 | 20.4 | 4.5 |
| $128^2$ | 230.9 | 35.2 | 6.6 |
| $129^2$ | 96.9 | 20.9 | 4.6 |
| $160^2$ | 179.3 | 34.4 | 5.2 |
| $255^2$ | 819.2 | 90.3 | 9.1 |
| $256^2$ | 2170.8 | 188.7 | 11.5 |
| $257^2$ | 871.7 | 93.3 | 9.3 |
| $511^2$ | 7977.2 | 458.8 | 17.4 |
| $512^2$ | 20269 | 739.3 | 27.4 |
| $513^2$ | 7887.2 | 461.9 | 17.1 |
| $600^2$ | 10841.4 | 644.2 | 16.8 |
| $700^2$ | 15048.8 | 1162.4 | 12.9 |
| $800^2$ | | 1936.6 | |

variations in the difficulty of the specific eigenvalue problem (eg. nearly multiple eigenvalues). The hierarchical representation has proven to be effective in a min-cut based segmentation algorithm that we proposed recently [8]. Here we explored the use of random walks and associated spectral embedding techniques for the automatic generation of suitable proposal (source and sink) regions for a min-cut based algorithm. The multi-scale algorithm was used to generate the 40 leading eigenvectors of large transition matrices (eg. size $20K \times 20K$). In terms of future work, it will be useful to compare our work with other approximate methods for SVD such as [23].

**Ack:** We thank S. Roweis, F. Estrada and M. Sakr for valuable comments.

## References

[1]  D. Achlioptas and F. McSherry. Fast Computation of Low-Rank Approximations. *STOC*, 2001.

[2]  D. Achlioptas *et al* Sampling Techniques for Kernel Methods. *NIPS*, 2001.

[3]  S. Barnard and H. Simon  Fast Multilevel Implementation of Recursive Spectral Bisection for Partitioning Unstructured Problems. PPSC, 627-632.

[4]  M. Belkin *et al* Laplacian Eigenmaps and Spectral Techniques for Embedding. *NIPS*, 2001.

[5]  M. Brand *et al* A unifying theorem for spectral embedding and clustering. *AI & STATS*, 2002.

[6]  C. Chennubhotla. Spectral Methods for Multi-scale Feature Extraction and Spectral Clustering. `http://www.cs.toronto.edu/~chakra/thesis.pdf` Ph.D Thesis, Department of Computer Science, University of Toronto, Canada, 2004.

[7]  C. Chennubhotla and A. Jepson. Half-Lives of EigenFlows for Spectral Clustering. *NIPS*, 2002.

[8]  F. Estrada, A. Jepson and C. Chennubhotla. Spectral Embedding and Min-Cut for Image Segmentation. *Manuscript Under Review*, 2004.

[9]  C. Fowlkes *et al* Efficient spatiotemporal grouping using the Nystrom method. *CVPR*, 2001.

[10]  S. Belongie *et al*  Spectral Partitioning with Indefinite Kernels using Nystrom app. *ECCV*, 2002.

[11]  A. Frieze *et al*  Fast Monte-Carlo Algorithms for finding low-rank approximations. *FOCS*, 1998.

[12]  Y. Koren *et al* ACE: A Fast Multiscale Eigenvectors Computation for Drawing Huge Graphs IEEE Symp. on InfoVis 2002, pp. 137-144

[13]  R. B. Lehoucq, D. C. Sorensen and C. Yang. ARPACK User Guide: Solution of Large Scale Eigenvalue Problems by Implicitly Restarted Arnoldi Methods. SIAM 1998.

[14]  J. J. Lin. Reduced Rank Approximations of Transition Matrices. *AI & STATS*, 2002.

[15]  L. Lova'sz. Random Walks on Graphs: A Survey Combinatorics, 1996, 353–398.

[16]  G. J. McLachlan *et al* Mixture Models: Inference and Applications to Clustering. 1988

[17]  M. Meila and J. Shi. A random walks view of spectral segmentation. *AI & STATS*, 2001.

[18]  A. Ng, M. Jordan and Y. Weiss. On Spectral Clustering: analysis and an algorithm *NIPS*, 2001.

[19]  A. Pothen  Graph partitioning algorithms with applications to scientific computing. Parallel Numerical Algorithms, D. E. Keyes et al (eds.), Kluwer Academic Press, 1996.

[20]  G. L. Scott *et al*  Feature grouping by relocalization of eigenvectors of the proximity matrix. *BMVC*, pg. 103-108, 1990.

[21]  E. Sharon *et al* Fast Multiscale Image Segmentation *CVPR*, I:70-77, 2000.

[22]  J. Shi and J. Malik. Normalized cuts and image segmentation. *PAMI*, August, 2000.

[23]  H. Simon *et al* Low-Rank Matrix Approximation Using the Lanczos Bidiagonalization Process with Applications SIAM J. of Sci. Comp. 21(6):2257-2274, 2000.

[24]  N. Tishby *et al* Data clustering by Markovian Relaxation *NIPS*, 2001.

[25]  C. Williams *et al* Using the Nystrom method to speed up the kernel machines. *NIPS*, 2001.
